# Interposing an ontogenic model between Genetic Algorithms and Neural Networks

Richard K. Belew
rik@cs.ucsd.edu

Cognitive Computer Science Research Group
Computer Science & Engr. Dept. (0014)
University of California - San Diego
La Jolla, CA 92093

## Abstract

The relationships between learning, development and evolution in Nature is taken seriously, to suggest a model of the developmental process whereby the genotypes manipulated by the Genetic Algorithm (GA) might be expressed to form phenotypic neural networks (NNet) that then go on to learn. ONTOL is a grammar for generating polynomial NNets for time-series prediction. Genomes correspond to an ordered sequence of ONTOL productions and define a grammar that is expressed to generate a NNet. The NNet's weights are then modified by learning, and the individual's prediction error is used to determine GA fitness. A new gene doubling operator appears critical to the formation of new genetic alternatives in the preliminary but encouraging results presented.

## 1 Introduction

Two natural phenomena, the learning done by individuals' nervous systems and the evolution done by populations of individuals, have served as the basis of distinct classes of adaptive algorithms, neural networks (NNets) and Genetic Algorithms (GAs), resp. Interactions between learning and evolution in Nature suggests that combining NNet and GA algorithmic techniques might also yield interesting hybrid algorithms.

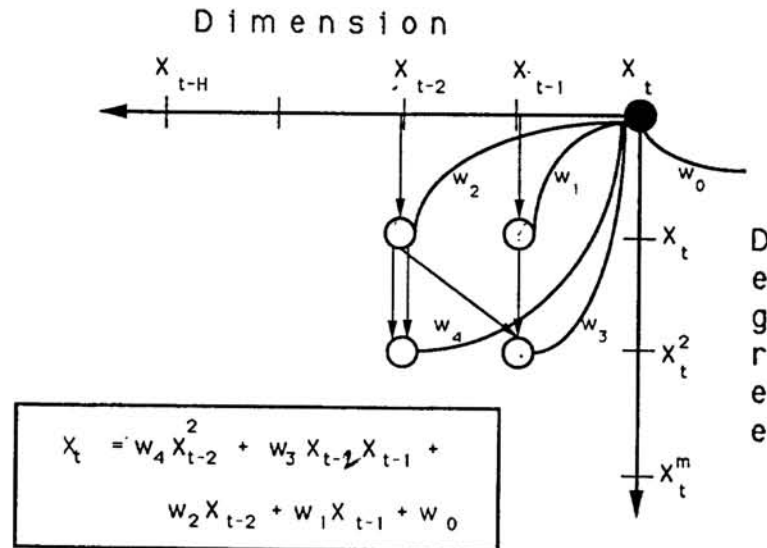

Figure 1: Polynomial networks

Taking the analogy to learning and evolution seriously, we propose that the missing feature is a the *developmental* process whereby the genotypes manipulated by the GA are *expressed* to form phenotypic NNets that then go on to learn. Previous attempts to use the GA to search for good NNet topologies have foundered exactly because they have assumed an overly direct genotype-to-phenotype correspondence. This research is therefore consistent with other NNet research the physiology of neural development [3] as well as those into "constructive" methods for changing network topologies adaptively during the training process [4]. Additional motivation derives from the growing body of neuroscience demonstrating the importance of developmental processes as the shapers of effective learning networks. Cognitively, the resolution of false dicotomies like "nature/nurture" and "nativist/empiricist" also depends on a richer language for describing the way genetically determined characteristics and within-lifetime changes by individuals can interact.

Because GAs and NNets are each complicated technologies in their own right, and because the focus of the current research is a model of development that can span between them, three major simplifications have been imposed for the preliminary research reported here. First, in order to stay close to the mathematical theory of functional approximation, we restrict the form of our NNets to what can be called "polynomials networks" (cf. [7]). That is, we will consider networks with a first layer of linear units (i.e., terms in the polynomial that are simply weighted input $X_i$), a second layer with units that form products of first-layer units, a third layer with units that form products of second-layer units, etc.; see Figure 1, and below for an example. As depicted in Figure 1 the space of polynomial networks can be viewed as two-dimensional, parameterized by *dimension* (i.e., how much history of the time series is used) and *degree*.

There remains the problem of finding the best parameter values for this particular polynomial form. Much of classicial optimization theory and more recent NNet research is concerned with various methods for performing this task. Previous re-

search has demonstrated that the *global sampling* behavior of the GA works very effectively with any gradient, *local search* technique [2]. The second major simplification, then, is that for the time being we use only the most simple-minded gradient method: first-order, fixed-step gradient descent. Analytically, this is the most tractable, and the general algorithm design can readily replace this with any other local search technique.

The final simplification is that we focus on one of the most parsimonious of problems, time series prediction: The GA is used to evolve NNets that are good at predicting $X_{t+1}$ given access to an unbounded history $X_t, X_{t-1}, X_{t-2}, ....$ Polynomial approximations of an arbitrary time series can vary in two dimensions: the extent to which they rely on this history, and (e.g., how far back in time), and in their degree. The Stone-Weierstrauss Aproximation Theorem guarantees that, within this two-dimensional space, there exists some polynomial that will match the desired temporal sequence to arbitrarily precision. The problem, of course, is that over a history $H$ and allowing $m$ degree terms there exists $O(H^m)$ terms, far too many to search effectively. From the perspective of function approximation, then, this work corresponds to a particular heuristic for searching for the correct polynomial form, the parameters of which will be tuned with a gradient technique.

## 2    Expression of the ONTOL grammar

Every multi-cellular organism has the problem of using a single genetic description contained in the first germ cell as specification for *all* of its various cell types. The genome therefore appears to contain a set of developmental instructions, subsets of which become "relevant" to the particular context in which each developing cell finds itself. If we imagine that each cell type is a unique symbol in some alphabet, and that the mature organism is a string of symbols, it becomes very natural to model the developmental process as a (context-sensitive) *grammar* generating this string [6, 5]. The initial germ cell becomes the start symbol. A series of production rules specify the expansion (mitosis) of this non-terminal (cell) into two other symbols that then develop according to the same set of genetically-determined rules, until all cells are in a mature, terminal state.

ONTOL is a grammar for generating cells in the two-dimensional space of polynomial networks. The left hand side (LHS) of productions in this grammar define conditions on the cells' internal *Clock* state and on the state of its eight Moore neighbors. The RHS of the production defines one of five cell-state update actions that are performed if the LHS condition is satisfied: A cell can mitosize either left or down (*MLeft, MDown*), meaning that this adjacent cell now becomes filled with an identical copy; *Die* (i.e., disappear entirely); *Tick* (simply decrement its internal *Clock* state); or *Terminate* (cease development). Only terminating cells form synaptic connections, and only to adjacent neighbors.

The developmental process is begun by placing a single "gamete" cell at the origin of the 2d polyspace, with its *Clock* state initialized to a maximal value $MaxClock = 4$; this state is decremented every time a gene is fired. If and when a gene causes this cell to undergo mitosis, a new cell, either to the left or below the originial cell, is created. Critically, the same set of genetic instructions contained in the original gametic cell are used to control transitions of *all* its progeny cells (much like a

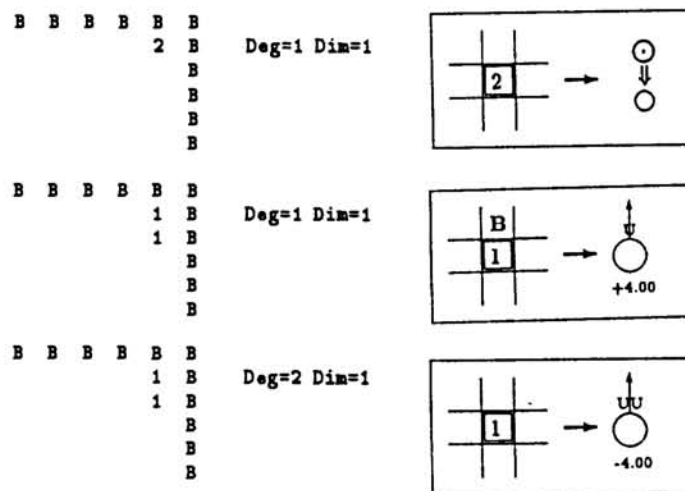

Figure 2: Logistic genome, engineered

cellular automaton's transition table), even though the differing contexts of each cell are likely to cause different genes to be applied in different cells. Figure 2 shows a trace of this developmental process: each snap-shot shows the *Clock* states of all active (non-terminated) cells, the coordinates of the cell being expressed, and the gene used to control its expression.

## 3   Experimental design

Each generation begins by developing and evaluating each genotype in the population. First, each genome in the population is expressed to form an executable Lisp lambda expression computing a polynomial and a corresponding set of initial weights for each of its terms. If this expression can be performed successfully and the individual is viable (i.e., their genomes can be interpreted to build well-formed networks), the individual is exposed to $NTrain$ sequential instances of the time series. Fitness is then defined to be its cumulative error on the next $NTest$ time steps.

After the entire population has been evaluated, the next generation is formed according to a relatively conventional genetic algorithm: more successful individuals are differentially reproduced and genetic operators are applied to these to experiment with novel, but similar, alternatives. Each genome is cloned zero, one or more times using a proportional selection algorithm that guarantees the expected number of offspring is proportional to an individual's relative fitness.

Variation is introduced into the population by *mutation* and *recombination* genetic operators that explore new genes and genomic combinations. Four types of mutation were applied, with the probability of a mutation proportional to genome length. First, some random portion of an extant gene might be randomly altered, e.g., changing an initial weight, adding or deleting a constraint on a condition, changing the gene's action. Because a gene's order in the genome can affect its probability of being expressed, a second form of mutation permutes the order of the genes on

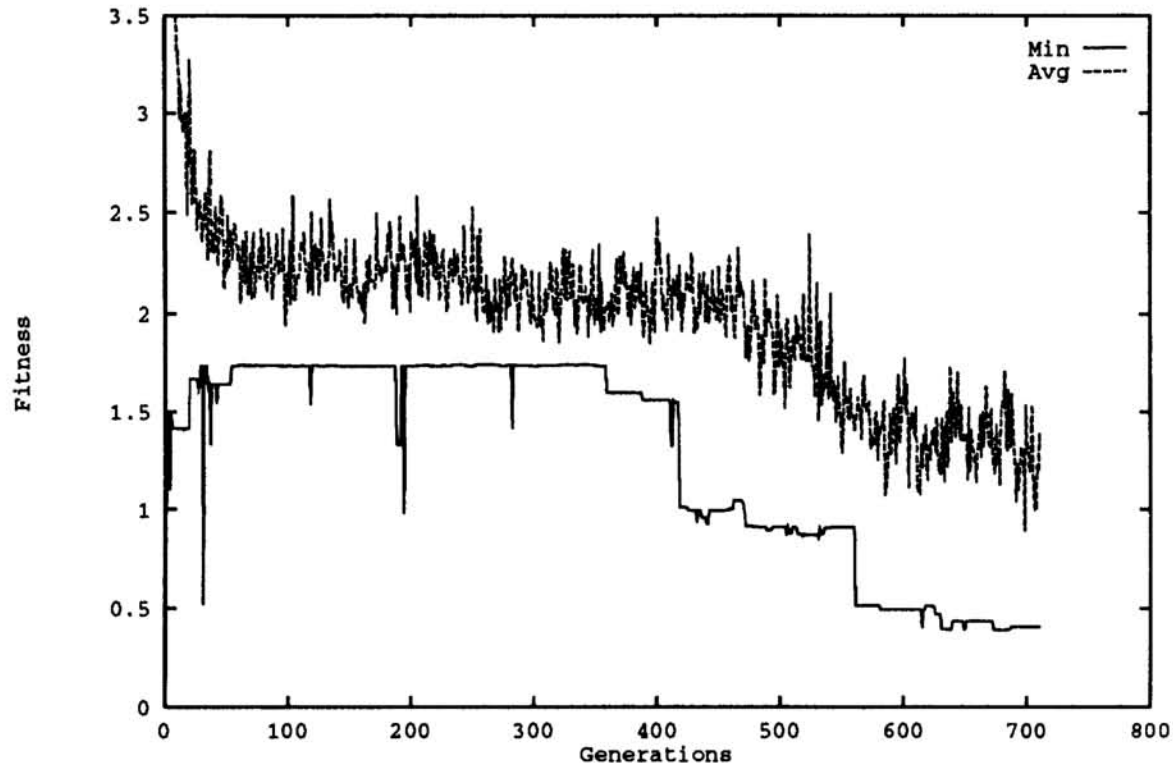

Figure 3: Poulation Minimum and Average Fitness

the genome. A third class of mutation removes genes from the genome, always "trimming" them from the end. Combined with the expression mechanism's bias towards the head of the genomic list, this trimming operation creates a pressure towards putting genes critical to early ontogeny near the head. The final and critical form of mutation randomly selects a gene to be *doubled*: a duplicate copy of the gene is constructed and inserted at a randomly selected position in the genome. After all mutations have been performed, cross-over is performed between pairs of individuals.

# 4    Experiments

To demonstrate, consider the problem of predicting a particularly difficult time series, the chaotic logistic map: $X_t = 4.0X_{t-1} - 4.0X_{t-1}^2$. The example of Figure 2 showed an ONTOL genome engineered to produce the desired logistic polynomial. This "genetically engineered" solution is merely evidence that a genetic solution exists that can be interpretted to form the desired phenotypic form; the real test is of course whether the GA can find it or something similar.

Early generations are not encouraging. Figure 3 shows the minimum (i.e., best) prediction error and population average error for the first 800 generations of a typical simulation. Initial progress is rapid because in the initial, randomly constructed population, fully half of the individuals are not even viable. These are strongly

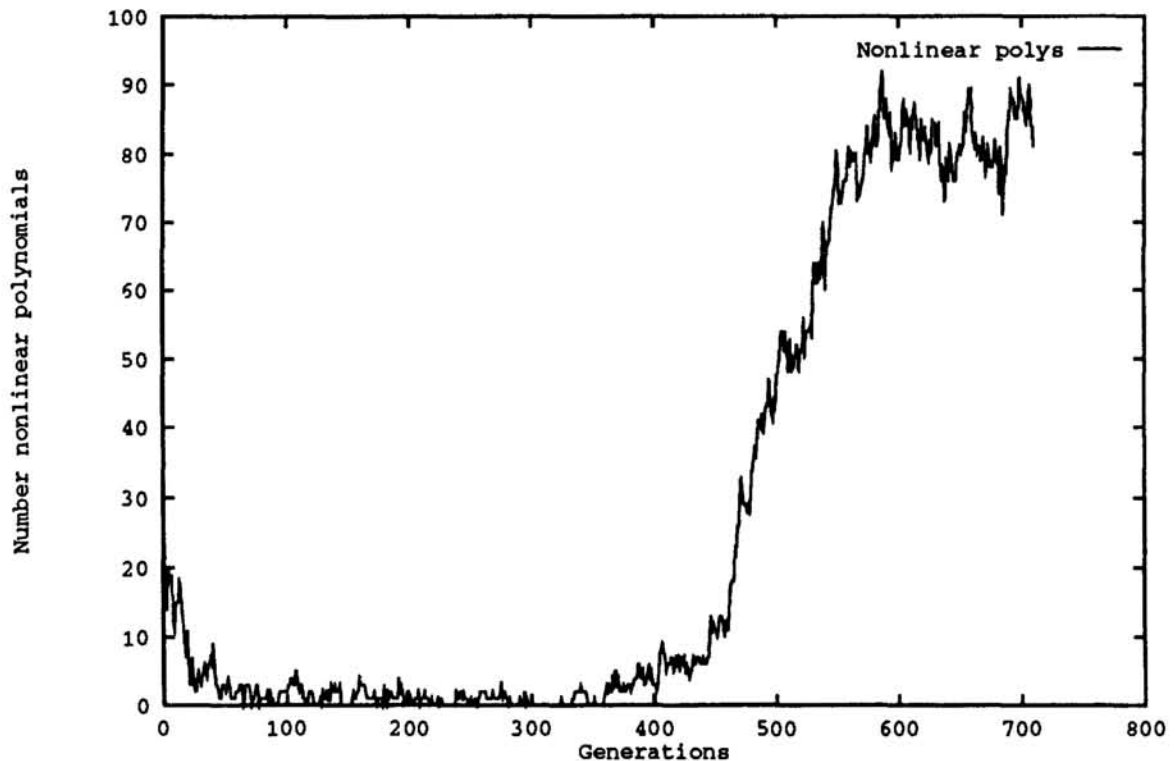

Figure 4: Complex polynomials

selected against, of course, and within the first two or three generations at least 95% of all generations remain viable.

For the next several hundred generations, however, all of ONTOL's developmental machinery appears for naught as the dominant phenotypic individuals are the most "simplistic" linear, first-degree approximators of the form $w_1 X_1 + w_0$. Even here, however, the GA is able to work in conjunction with the gradient learning process is able to achieve Baldwin-like effects optimizing $w_0$ and $w_1$ [1]. The simulation reaches a "simplistic plateau," then, as it converges on a population composed of the best predictors the simplistic linear, first-degree network topology permits for this time series.

In the background, however, genetic operators are continuing to explore a wide variety of *genotypic* forms that all have the property of generating roughly the same simplistic *phenotypes*. Figure 4 shows that there are significant numbers of "complex" polynomials[1] in early generations, and some of these have much higher than average fitness[2] On average, however, genes leading to complex phenotypes provide lead to poorer approximations than the simplistic ones, and are quickly culled.

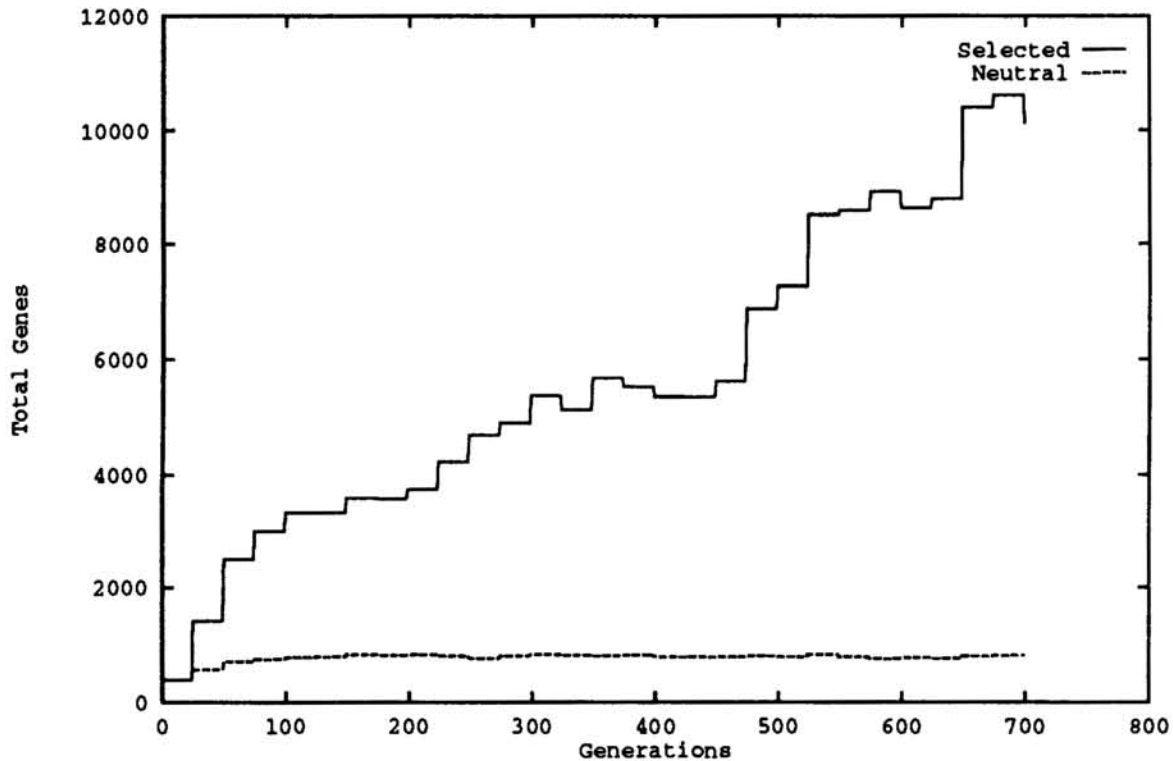

Figure 5: Genome length

A critical aspect of the redundancy introduced by gene doubling is that old genetic material is freed to mutate into new forms without threatening the phenotype's viability. When compared to a population of mediocre, simplistic networks any complex networks able to provide more accurate predictions have much higher fitness, and eventually are able to take over the population. Around generation 400, then, Figure 3 shows the fitness dropping from the simplistic plateau, and Figure 4 shows the number of complex polynomials increasing. Many of these individuals' genomes indeed encode grammars that form polynomials of the desired functional form.

A surprising feature of these simulations is that while the genes leading to complex phenotypes are present from the beginning and continue to be explored during the simplistic plateau, it takes many generations before these genes are successfully composed into robust, consistently viable genotypes. How do the complex genotypes discovered in later generations differ from those in the initial population?

One piece of the answer is revealed in Figure 5: later genomes are much longer. All 100 individuals in the initial population have exactly five genes, and so the initial "gene pool" size is 500. In the experiments just described, this number grows asymptotically to approximately 6000 total genes (i.e., 60 per individual, on average) during the simplistic plateau, and then explodes a second time to more than 10,000 as the population converts to complex polynomials. It appears that gene duplication creates a very constructive form of redundancy: mulitple copies of crit-

ical genes help the genotype maintain the more elaborate development programs required to form complex phenotypes. Micro-analysis of the most successful individuals in later generations supports this view. While many parts of their genomes appear inconsequential (for example, relative to the engineered genome of Figure 2), both the *M Down* gene and the two-element *Terminate* genes, critical to forming polynomials that are "morphologically isomorphic" with the correct solution, are consistently present.

This hypothesis is also supported by results from a second experiment, also plotted on Figure 5. Recall that the increase in genome size caused by gene doubling is offset by a trimming mutation that periodically shortens a genome. The curve labelled "Neutral" shows the results of these opposing operations when the next generation is formed randomly, rather than being selected for better prediction. Under neutral selection, genome size grows slightly from initial size, but gene doubling and genome trimming then quickly reach equilibrium. When we select for better predictors, however, longer genomes are clearly preferred, at least up to a point. The apparent asymptote accompanying the simplistic plateau suggests that if these simulations were extended, the length of complex genotypes would also stabalize.a

## Acknowledgements

I gratefully acknowledge the warm and stimulating research environments provide by Domenico Parisi and colleagues at the Psychological Institute, CNR, Rome, Italy, and Jean-Arcady Meyer and colleagues in the Groupe de BioInformatique, Ecole Normale Superieure in Paris, France.

## Footnotes

[1] I.e., either nonlinear terms or higher dimensional dependence on the past

[2] Note the good solutions in the first 50 generations, as well as subsequent dips during the simplistic plateau.

## References

[1] R. K. Belew. Evolution, learning and culture: computational metaphors for adaptive search. *Complex Systems*, 4(1):11–49, 1990.

[2] R. K. Belew, J. McInerney, and N. N. Schraudolph. Evolving networks: Using the Genetic Algorithm with connectionist learning. In *Proc. Second Artificial Life Conference*, pages 511–547, New York, 1991. Addison-Wesley.

[3] J. D. Cowan and A. E. Friedman. Development and regeneration of eye-brain maps: A computational model. In *Advances in Neural Info. Proc. Systems 2*, pages 92–99. Morgan Kaufman, 1990.

[4] S. E. Fahlman and C. Lebiere. The Cascade-Correlation learning architecture. In D. S. Touretzky, editor, *Advances in Neural Info. Proc. Systems 4*, pages 524–532. Morgan Kaufmann, 1990.

[5] H. Kitano. Designing neural networks using genetic algorithms with graph generation system. *Complex Systems*, 4(4), 1990.

[6] A. Lindenmayer and G. Rozenberg. *Automata, languages, development*. North-Holland, Amsterdam, 1976.

[7] T. D. Sanger, R. S. Sutton, and C. J. Matheus. Iterative construction of sparse polynomials. In J. E. Moody, S. J. Hanson, and R. P. Lippman, editors, *Advances in Neural Info. Proc. Systems 4*, pages 1064–1071. Morgan Kaufmann, 1992.